# Natural Image Denoising with Convolutional Networks

**Viren Jain**[1]
[1]Brain & Cognitive Sciences
Massachusetts Institute of Technology

**H. Sebastian Seung**[1,2]
[2]Howard Hughes Medical Institute
Massachusetts Institute of Technology

## Abstract

We present an approach to low-level vision that combines two main ideas: the use of convolutional networks as an image processing architecture and an unsupervised learning procedure that synthesizes training samples from specific noise models. We demonstrate this approach on the challenging problem of natural image denoising. Using a test set with a hundred natural images, we find that convolutional networks provide comparable and in some cases superior performance to state of the art wavelet and Markov random field (MRF) methods. Moreover, we find that a convolutional network offers similar performance in the blind denoising setting as compared to other techniques in the non-blind setting. We also show how convolutional networks are mathematically related to MRF approaches by presenting a mean field theory for an MRF specially designed for image denoising. Although these approaches are related, convolutional networks avoid computational difficulties in MRF approaches that arise from probabilistic learning and inference. This makes it possible to learn image processing architectures that have a high degree of representational power (we train models with over 15,000 parameters), but whose computational expense is significantly less than that associated with inference in MRF approaches with even hundreds of parameters.

## 1 Background

Low-level image processing tasks include edge detection, interpolation, and deconvolution. These tasks are useful both in themselves, and as a front-end for high-level visual tasks like object recognition. This paper focuses on the task of denoising, defined as the recovery of an underlying image from an observation that has been subjected to Gaussian noise.

One approach to image denoising is to transform an image from pixel intensities into another representation where statistical regularities are more easily captured. For example, the Gaussian scale mixture (GSM) model introduced by Portilla and colleagues is based on a multiscale wavelet decomposition that provides an effective description of local image statistics [1, 2].

Another approach is to try and capture statistical regularities of pixel intensities directly using Markov random fields (MRFs) to define a prior over the image space. Initial work used hand-designed settings of the parameters, but recently there has been increasing success in learning the parameters of such models from databases of natural images [3, 4, 5, 6, 7, 8]. Prior models can be used for tasks such as image denoising by augmenting the prior with a noise model.

Alternatively, an MRF can be used to model the probability distribution of the clean image conditioned on the noisy image. This conditional random field (CRF) approach is said to be discriminative, in contrast to the generative MRF approach. Several researchers have shown that the CRF approach can outperform generative learning on various image restoration and labeling tasks [9, 10]. CRFs have recently been applied to the problem of image denoising as well [5].

The present work is most closely related to the CRF approach. Indeed, certain special cases of convolutional networks can be seen as performing maximum likelihood inference on a CRF [11]. The advantage of the convolutional network approach is that it avoids a general difficulty with applying MRF-based methods to image analysis: the computational expense associated with both parameter estimation and inference in probabilistic models. For example, naive methods of learning MRF-based models involve calculation of the partition function, a normalization factor that is generally intractable for realistic models and image dimensions. As a result, a great deal of research has been devoted to approximate MRF learning and inference techniques that meliorate computational difficulties, generally at the cost of either representational power or theoretical guarantees [12, 13].

Convolutional networks largely avoid these difficulties by posing the computational task within the statistical framework of $regression$ rather than density estimation. Regression is a more tractable computation and therefore permits models with greater representational power than methods based on density estimation. This claim will be argued for with empirical results on the denoising problem, as well as mathematical connections between MRF and convolutional network approaches.

## 2 Convolutional Networks

Convolutional networks have been extensively applied to visual object recognition using architectures that accept an image as input and, through alternating layers of convolution and subsampling, produce one or more output values that are thresholded to yield binary predictions regarding object identity [14, 15]. In contrast, we study networks that accept an image as input and produce an entire image as output. Previous work has used such architectures to produce images with binary targets in image restoration problems for specialized microscopy data [11, 16]. Here we show that similar architectures can also be used to produce images with the analog fluctuations found in the intensity distributions of natural images.

**Network Dynamics and Architecture**

A convolutional network is an alternating sequence of linear filtering and nonlinear transformation operations. The input and output layers include one or more images, while intermediate layers contain "hidden" units with images called feature maps that are the internal computations of the algorithm. The activity of feature map $a$ in layer $k$ is given by

$$I_{k,a} = f\left(\sum_b w_{k,ab} \otimes I_{k-1,b} - \theta_{k,a}\right) \quad (1)$$

where $I_{k-1,b}$ are feature maps that provide input to $I_{k,a}$, and $\otimes$ denotes the convolution operation. The function $f$ is the sigmoid $f(x) = 1/\left(1 + e^{-x}\right)$ and $\theta_{k,a}$ is a bias parameter.

We restrict our experiments to monochrome images and hence the networks contain a single image in the input layer. It is straightforward to extend this approach to color images by assuming an input layer with multiple images (e.g., RGB color channels). For numerical reasons, it is preferable to use input and target values in the range of 0 to 1, and hence the 8-bit integer intensity values of the dataset (values from 0 to 255) were normalized to lie between 0 and 1. We also explicitly encode the border of the image by padding an area surrounding the image with values of $-1$.

**Learning to Denoise**

Parameter learning can be performed with a modification of the backpropagation algorithm for feedfoward neural networks that takes into account the weight-sharing structure of convolutional networks [14]. However, several issues have to be addressed in order to learn the architecture in Figure 1 for the task of natural image denoising.

Firstly, the image denoising task must be formulated as a learning problem in order to train the convolutional network. Since we assume access to a database of only clean, noiseless images, we implicitly specify the desired image processing task by integrating a noise process into the training procedure. In particular, we assume a noise process $n(x)$ that operates on an image $x_i$ drawn from a distribution of natural images $X$. If we consider the entire convolutional network to be some function

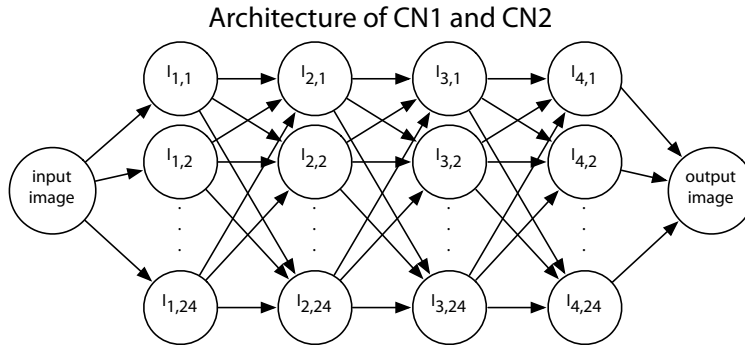

Figure 1: Architecture of convolutional network used for denoising. The network has 4 hidden layers and 24 feature maps in each hidden layer. In layers 2, 3, and 4, each feature map is connected to 8 randomly chosen feature maps in the previous layer. Each arrow represents a single convolution associated with a $5 \times 5$ filter, and hence this network has 15,697 free parameters and requires 624 convolutions to process its forward pass.

$F_\phi$ with free parameters $\phi$, then the parameter estimation problem is to minimize the reconstruction error of the images subject to the noise process: $\min_\phi \sum_i (x_i - F_\phi(n(x_i)))^2$.

Secondly, it is inefficient to use batch learning in this context. The training sets used in the experiments have millions of pixels, and it is not practical to perform both a forward and backward pass on the entire training set when gradient learning requires many tens of thousands of updates to converge to a reasonable solution. Stochastic online gradient learning is a more efficient learning procedure that can be adapted to this problem. Typically, this procedure selects a small number of independent examples from the training set and averages together their gradients to perform a single update. We compute a gradient update from $6 \times 6$ patches randomly sampled from six different images in the training set. Using a localized image patch violates the independence assumption in stochastic online learning, but combining the gradient from six separate images yields a $6 \times 6 \times 6$ cube that in practice is a sufficient approximation of the gradient to be effective. Larger patches (we tried $8 \times 8$ and $10 \times 10$) reduce correlations in the training sample but do not improve accuracy. This scheme is especially efficient because most of the computation for a local patch is shared.

We found that training time is minimized and generalization accuracy is maximized by incrementally learning each layer of weights. Greedy, layer-wise training strategies have recently been explored in the context of unsupervised initialization of multi-layer networks, which are usually fine tuned for some discriminative task with a different cost function [17, 18, 19]. We maintain the same cost function throughout. This procedure starts by training a network with a single hidden layer. After thirty epochs, the weights from the first hidden layer are copied to a new network with two hidden layers; the weights connecting the hidden layer to the output layer are discarded. The two hidden layer network is optimized for another thirty epochs, and the procedure is repeated for *N* layers.

Finally, when learning networks with two or more hidden layers it was important to use a very small learning rate for the final layer (0.001) and a larger learning rate (0.1) in all other layers.

**Implementation**

Convolutional network inference and learning can be implemented in just a few lines of MATLAB code using multi-dimensional convolution and cross-correlation routines. This also makes the approach especially easy to optimize using parallel computing or GPU computing strategies.

## 3 Experiments

We derive training and test sets for our experiments from natural images in the Berkeley segmentation database, which has been previously used to study denoising [20, 4]. We restrict our experiments to the case of monochrome images; color images in the Berkeley dataset are converted to grayscale by averaging the color channels. The test set consists of 100 images, 77 with dimensions $321 \times 481$ and 23 with dimensions $481 \times 321$. Quantitative comparisons are performed using the Peak Signal

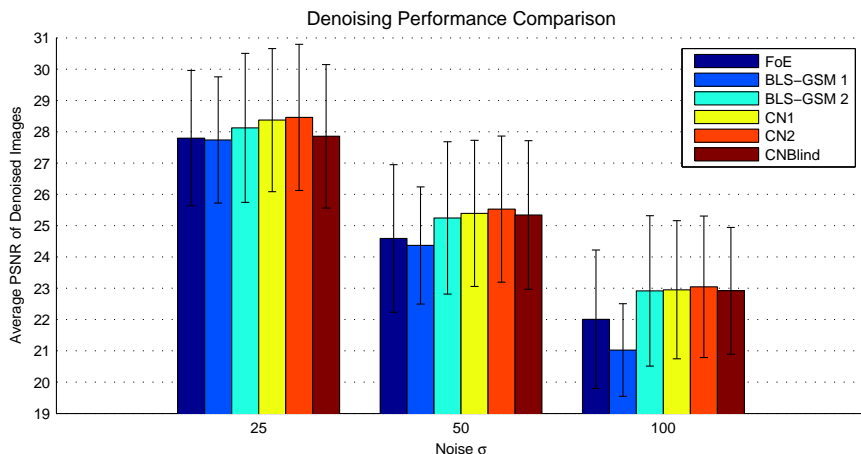

Figure 2: Denoising results as measured by peak signal to noise ratio (PSNR) for 3 different noise levels. In each case, results are the average denoised PSNR of the hundred images in the test set. CN1 and CNBlind are learned using the same forty image training set as the Field of Experts model (FoE). CN2 is learned using a training set with an additional sixty images. BLS-GSM1 and BLS-GSM2 are two different parameter settings of the algorithm in [1]. All methods except CNBlind assume a known noise distribution.

to Noise Ratio (PSNR): $20 \log_{10}(255/\sigma_e)$, where $\sigma_e$ is the standard deviation of the error. PSNR has been widely used to evaluate denoising performance [1, 4, 2, 5, 6, 7].

**Denoising with known noise conditions**

In this task it is assumed that images have been subjected to Gaussian noise of known variance. We use this noise model during the training process and learn a five-layer network for each noise level. Both the Bayes Least Squares-Gaussian Scale Mixture (BLS-GSM) and Field of Experts (FoE) method also optimize the denoising process based on a specified noise level.

We learn two sets of networks for this task that differ in their training set. In one set of networks, which we refer to as CN1, the training set is the same subset of the Berkeley database used to learn the FoE model [4]. In another set of networks, called CN2, this training set is augmented by an additional sixty images from the Berkeley database. The architecture of these networks is shown in Fig. 1. Quantitative results from both networks under three different noise levels are shown in Fig. 2, along with results from the FoE and BLS-GSM method (BLS-GSM 1 is the same settings used in [1] while BLS-GSM 2 is the default settings in the code provided by the authors). For the FoE results, the number of iterations and magnitude of the step size are optimized for each noise level using a grid search on the training set. A visual comparison of these results is shown in Fig. 3.

We find that the convolutional network has the highest average PSNR using either training set, although by a margin that is within statistical insignificance when standard error is computed from the distribution of PSNR values of the entire image. However, we believe this is a conservative estimate of the standard error, which is much smaller when measured on a pixel or patch-wise basis.

**Blind denoising**

In this task it is assumed that images have been subjected to Gaussian noise of *unknown* variance. Denoising in this context is a more difficult problem than in the non-blind situation. We train a single six-layer network network we refer to as CNBlind by randomly varying the amount of noise added to each example in the training process, in the range of $\sigma = [0, 100]$ . During inference, the noise level is unknown and only the image is provided as input. We use the same training set as the FoE model and CN1. The architecture is the same as that shown in Fig. 1 except with 5 hidden layers instead of 4. Results for 3 noise levels are shown in Fig. 2. We find that a convolutional network trained for blind denoising performs well even compared to the other methods under non-blind conditions. In Fig. 4, we show filters that were learned for this network.

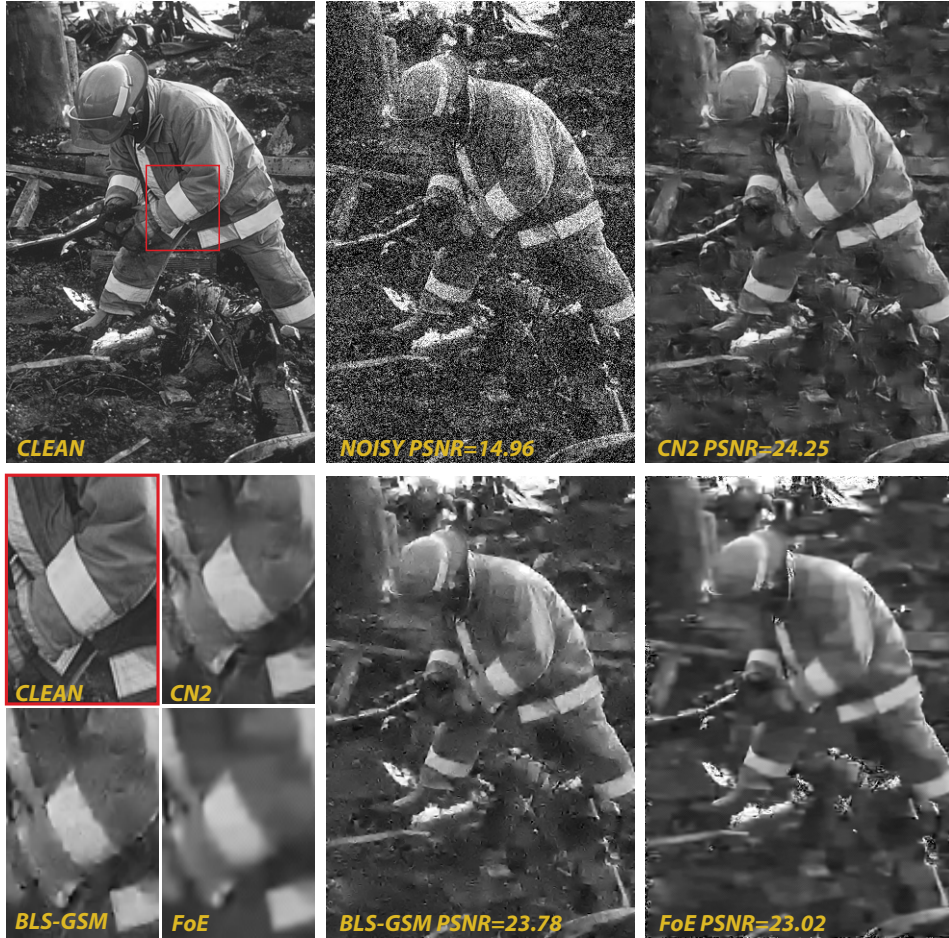

Figure 3: Denoising results on an image from the test set. The noisy image was generated by adding Gaussian noise with $\sigma = 50$ to the clean image. Non-blind denoising results for the BLS-GSM, FoE, and convolutional network methods are shown. The lower left panel shows results for the outlined region in the upper left panel. The zoomed in region shows that in some areas CN2 output has less severe artifacts than the wavelet-based results and is sharper than the FoE results. CN1 results (PSNR=24.12) are visually similar to those of CN2.

## 4  Relationship between MRF and Convolutional Network Approaches

In the introduction, we claim that convolutional networks have similar or even greater representational power compared to MRFs. To support this claim, we will show that special cases of convolutional networks correspond to mean field inference for an MRF. This does not rigorously prove that convolutional networks have representational power greater than or equal to MRFs, since mean field inference is an approximation. However, it is plausible that this is the case.

Previous work has pointed out that the Field of Experts MRF can be interpreted as a convolutional network (see [21]) and that MRFs with an Ising-like prior can be related to convolutional networks (see [11]). Here, we analyze a different MRF that is specially designed for image denoising and show that it is closely related to the convolutional network in Figure 1. In particular, we consider an MRF that defines a distribution over analog "visible" variables $v$ and binary "hidden" variables $h$:

$$P(v, h) = \frac{1}{Z} \exp\left( -\frac{1}{2\sigma^2} \sum_i v_i^2 + \frac{1}{\sigma^2} \sum_{ia} h_i^a (w^a \otimes v)_i + \frac{1}{2} \sum_{iab} h_i^a (w^{ab} \otimes h^b)_i \right) \qquad (2)$$

where $v_i$ and $h_i$ correspond to the $i$th pixel location in the image, $Z$ is the partition function, and $\sigma$ is the known standard deviation of the Gaussian noise. Note that by symmetry we have $w_{i-j}^{ab} = w_{j-i}^{ba}$,

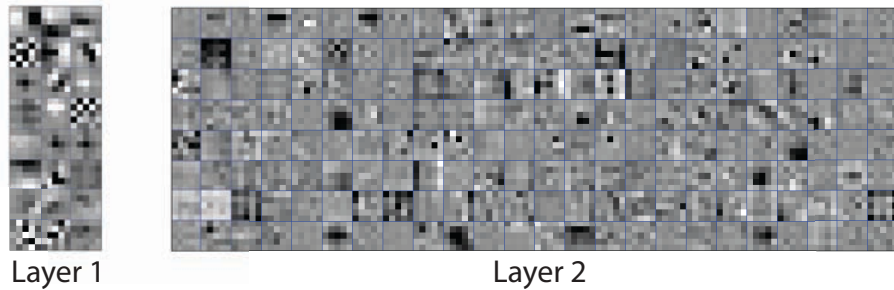

Layer 1                                        Layer 2

Figure 4: Filters learned for the first 2 hidden layers of network CNBlind. The second hidden layer has 192 filters (24 feature maps 8 filters per map). The first layer has recognizable structure in the filters, including both derivative filters as well as high frequency filters similar to those learned by the FoE model [4, 6].

and we assume $w_0^{aa} = 0$ so there is no self interaction in the model (if this were not the case, one could always transfer this to a term that is linear in $h_i^a$, which would lead to an additional bias term in the mean field approximation). Hence, $P(v \ h)$ constitutes an undirected graphical model which can be conceptualized as having separate layers for the visible and hidden variables. There are no intralayer interactions in the visible layer and convolutional structure (instead of full connectivity) in the intralayer interactions between hidden variables and interlayer interactions between the visible and hidden layer.

From the definition of $P(v \ h)$ it follows that the conditional distribution,

$$P(v \ h) \quad \exp \quad \frac{1}{2 \ ^2} \quad _i \quad v_i \quad _a (w^a \ h^a)_i \quad ^2 \tag{3}$$

is Gaussian with mean $v_i = \ _a (w^a \ h^a)_i$. This is also equal to the conditional expectation $E[v \ h]$. We can use this model for denoising by fixing the visible variables to the noisy image, computing the most likely hidden variables $h$ by MAP inference, and regarding the conditional expectation of $P(v \ h \ )$ as the denoised image. To do inference we would like to calculate $\max_h P(h \ v)$, but this is difficult because of the partition function. However, we can consider the mean field approximation,

$$h_i^a = f \quad \frac{1}{^2}(w^a \ v)_i + \ _b (w^{ab} \ h^b)_i \tag{4}$$

which can be solved by regarding the equation as a dynamics and iterating it. If we compare this to Eq. 1, we find that this is equivalent to a convolutional network in which each hidden layer has the same weights and each feature map directly receives input from the image.

These results suggest that certain convolutional networks can be interpreted as performing approximate inference on MRF models designed for denoising. In practice, the convolutional network architectures we train are not exactly related to such MRF models because the weights of each hidden layer are not constrained to be the same, nor is the image an input to any feature map except those in the first layer. An interesting question for future research is how these additional architectural constraints would affect performance of the convolutional network approach.

Finally, although the special case of non-blind Gaussian denoising allows for direct integration of the noise model into the MRF equations, our empirical results on blind denoising suggest that the convolutional network approach is adaptable to more general and complex noise models when specified implicitly through the learning cost function.

## 5   Discussion

**Prior versus learned structure**

Before learning, the convolutional network has little structure specialized to natural images. In contrast, the GSM model uses a multi-scale wavelet representation that is known for its suitability in

representing natural image statistics. Moreover, inference in the FoE model uses a procedure similar to non-linear diffusion methods, which have been previously used for natural image processing without learning. The architecture of the FoE MRF is so well chosen that even random settings of the free parameters can provide impressive performance [21].

Random parameter settings of the convolutional networks do not produce any clearly useful computation. If the parameters of CN2 are randomized in *just the last layer,* denoising performance for the image in Fig. 3 drops from PSNR=24.25 to 14.87. Random parameters in all layers yields even worse results. This is consistent with the idea that nothing in CN2's representation is specialized to natural images before training, other than the localized receptive field structure of convolutions. Our approach instead relies on a gradient learning algorithm to tune thousands of parameters using examples of natural images. One might assume this approach would require vastly more training data than other methods with more prior structure. However, we obtain good generalization performance using the same training set as that used to learn the Field of Experts model, which has many fewer degrees of freedom. The disadvantage of this approach is that it produces an architecture whose performance is more difficult to understand due to its numerous free parameters. The advantage of this approach is that it may lead to more accurate performance, and can be applied to novel forms of imagery that have very different statistics than natural images or any previously studied dataset (an example of this is the specialized image restoration problem studied in [11]).

**Network architecture and using more image context**

The amount of image context the convolutional network uses to produce an output value for a specific image location is determined by the number of layers in the network and size of filter in each layer. For example, the 5 and 6-layer networks explored here respectively use a $20 \times 20$ and $24 \times 24$ image patch. This is a relatively small amount of context compared to that used by the FoE and BLS-GSM models, both of which permit correlations to extend over the entire image. It is surprising that despite this major difference, the convolutional network approach still provides good performance. One explanation could be that the scale of objects in the chosen image dataset may allow for most relevant information to be captured in a relatively small field of view.

Nonetheless, it is of interest for denoising as well as other applications to increase the amount of context used by the network. A simple strategy is to further increase the number of layers; however, this becomes computationally intensive and may be an inefficient way to exploit the multi-scale properties of natural images. Adding additional machinery in the network architecture may work better. Integrating the operations of sub-sampling and super-sampling would allow a network to process the image at multiple scales, while still being entirely amenable to gradient learning.

**Computational efficiency**

With many free parameters, convolutional networks may seem like a computationally expensive image processing architecture. On the contrary, the 5-layer CN1 and CN2 architecture (Fig. 1) requires only 624 image convolutions to process an image. In comparison, the FoE model performs inference by means of a dynamic process that can require several thousand iterations. One-thousand iterations of these dynamics requires 48,000 convolutions (for an FoE model with 24 filters).

We also report wall-clock speed by denoising a $512 \times 512$ pixel image on a 2.16Ghz Intel Core 2 processor. Averaged over 10 trials, CN1/CN2 requires $38.86 \pm 0.1$ sec., 1,000 iterations of the FoE requires $1664.35 \pm 30.23$ sec. (using code from the authors of [4]), the BLS-GSM model with parameter settings "1" requires $51.86 \pm 0.12$ sec., and parameter setting "2" requires $26.51 \pm 0.15$ sec. (using code from the authors of [1]). All implementations are in MATLAB.

It is true, however, that training the convolutional network architecture requires substantial computation. As gradient learning can require many thousands of updates to converge, training the denoising networks required a parallel implementation that utilized a dozen processors for a week. While this is a significant amount of computation, it can be performed off-line.

**Learning more complex image transformations and generalized image attractors models**

In this work we have explored an image processing task which can be easily formulated as a learning problem by synthesizing training examples from abundantly available noiseless natural images. Can

this approach be extended to tasks in which the noise model has a more variable or complex form? Our results on blind denoising, in which the amount of noise may vary from little to severe, provides some evidence that it can. Preliminary experiments on image inpainting are also encouraging.

That said, a major virtue of the image prior approach is the ability to easily reuse a single image model in novel situations by simply augmenting the prior with the appropriate observation model. This is possible because the image prior and the observation model are decoupled. Yet explicit probabilistic modeling is computationally difficult and makes learning even simple models challenging. Convolutional networks forgo probabilistic modeling and, as developed here, focus on specific image to image transformations as a regression problem. It will be interesting to combine the two approaches to learn models that are "unnormalized priors" in the sense of energy-based image attractors; regression can then be used as a tool for unsupervised learning by capturing dependencies between variables within the same distribution [22].

**Acknowledgements**: we are grateful to Ted Adelson, Ce Liu, Srinivas Turaga, and Yair Weiss for helpful discussions. We also thank the authors of [1] and [4] for making code available.

# References

[1] J. Portilla, V. Strela, M.J. Wainwright, E.P. Simoncelli. Image denoising using scale mixtures of Gaussians in the wavelet domain. *IEEE Trans. Image Proc.*, 2003.

[2] S. Lyu, E.P. Simoncelli. Statistical modeling of images with fields of Gaussian scale mixtures. *NIPS* 2006*.

[3] S. Geman, D. Geman. Stochastic relaxation, Gibbs distributions and the Bayesian restoration of images. *Pattern Analysis and Machine Intelligence*, 1984.

[4] S. Roth, M.J. Black. Fields of Experts: a framework for learning image priors. *CVPR 2005*.

[5] M.F. Tappen, C. Liu, E.H. Adelson, W.T. Freeman. Learning Gaussian Conditional Random Fields for Low-Level Vision. *CVPR 2007*.

[6] Y. Weiss, W.T. Freeman. What makes a good model of natural images? *CVPR 2007*.

[7] P. Gehler, M. Welling. Product of "edge-perts". *NIPS* 2005*.

[8] S.C. Zhu, Y. Wu, D. Mumford. Filters, Random Fields and Maximum Entropy (FRAME): Towards a Unified Theory for Texture Modeling. *International Journal of Computer Vision*, 1998.

[9] S. Kumar, M. Hebert. Discriminative fields for modeling spatial dependencies in natural images. *NIPS* 2004*.

[10] X. He, R Zemel, M.C. Perpinan. Multiscale conditional random fields for image labeling. *CVPR 2004*.

[11] V. Jain, J.F. Murray, F. Roth, S. Turaga, V. Zhigulin, K.L. Briggman, M.N. Helmstaedter, W. Denk, H.S. Seung. Supervised Learning of Image Restoration with Convolutional Networks. *ICCV 2007*.

[12] S. Parise, M. Welling. Learning in markov random fields: An empirical study. *Joint Stat. Meeting*, 2005.

[13] R. Szeliski, R. Zabih, D. Scharstein, O. Veksler, V. Kolmogorov, A. Agarwala, M. Tappen, C. Rother. A comparative study of energy minimization methods for markov random fields. *ECCV 2006*.

[14] Y. LeCun, B. Boser, J.S. Denker, D. Henderson, R.E. Howard, W. Hubbard, L.D. Jackel. Backpropagation Applied to Handwritten Zip Code Recognition. *Neural Computation*, 1989.

[15] Y. LeCun, F.J. Huang, L. Bottou. Learning methods for generic object recognition with invariance to pose and lighting. *CVPR 2004*.

[16] F. Ning, D. Delhomme, Y. LeCun, F. Piano, L. Bottou, P.E. Barbano. Toward Automatic Phenotyping of Developing Embryos From Videos. *IEEE Trans. Image Proc.*, 2005.

[17] G. Hinton, R. Salakhutdinov. Reducing the dimensionality of data with neural networks. *Science*, 2006.

[18] M. Ranzato, YL Boureau, Y. LeCun. Sparse feature learning for deep belief networks. *NIPS* 2007*.

[19] Y. Bengio, P. Lamblin, D. Popovici, H. Larochelle. Greedy Layer-Wise Training of Deep Networks. *NIPS* 2006*.

[20] D. Martin, C. Fowlkes, D. Tal, J. Malik. A database of human segmented natural images and its application to evaluating segmentation algorithms and measuring ecological statistics. *ICCV 2001*.

[21] S. Roth. High-order markov random fields for low-level vision. PhD Thesis, Brown Univ., 2007.

[22] H.S. Seung. Learning continuous attractors in recurrent networks. *NIPS* 1997*.

